# Real-Time Control of a Tokamak Plasma Using Neural Networks

**Chris M Bishop**
Neural Computing Research Group
Department of Computer Science
Aston University
Birmingham, B4 7ET, U.K.
c.m.bishop@aston.ac.uk

**Paul S Haynes, Mike E U Smith, Tom N Todd,**
**David L Trotman and Colin G Windsor**
AEA Technology, Culham Laboratory,
Oxfordshire OX14 3DB
(Euratom/UKAEA Fusion Association)

## Abstract

This paper presents results from the first use of neural networks for the real-time feedback control of high temperature plasmas in a tokamak fusion experiment. The tokamak is currently the principal experimental device for research into the magnetic confinement approach to controlled fusion. In the tokamak, hydrogen plasmas, at temperatures of up to 100 Million K, are confined by strong magnetic fields. Accurate control of the position and shape of the plasma boundary requires real-time feedback control of the magnetic field structure on a time-scale of a few tens of microseconds. Software simulations have demonstrated that a neural network approach can give significantly better performance than the linear technique currently used on most tokamak experiments. The practical application of the neural network approach requires high-speed hardware, for which a fully parallel implementation of the multilayer perceptron, using a hybrid of digital and analogue technology, has been developed.

# 1   INTRODUCTION

Fusion of the nuclei of hydrogen provides the energy source which powers the sun. It also offers the possibility of a practically limitless terrestrial source of energy. However, the harnessing of this power has proved to be a highly challenging problem. One of the most promising approaches is based on magnetic confinement of a high temperature ($10^7 - 10^8$ Kelvin) plasma in a device called a tokamak (from the Russian for 'toroidal magnetic chamber') as illustrated schematically in Figure 1. At these temperatures the highly ionized plasma is an excellent electrical conductor, and can be confined and shaped by strong magnetic fields. Early tokamaks had plasmas with circular cross-sections, for which feedback control of the plasma position and shape is relatively straightforward. However, recent tokamaks, such as the COMPASS experiment at Culham Laboratory, as well as most next-generation tokamaks, are designed to produce plasmas whose cross-sections are strongly noncircular. Figure 2 illustrates some of the plasma shapes which COMPASS is designed to explore. These novel cross-sections provide substantially improved energy confinement properties and thereby significantly enhance the performance of the tokamak.

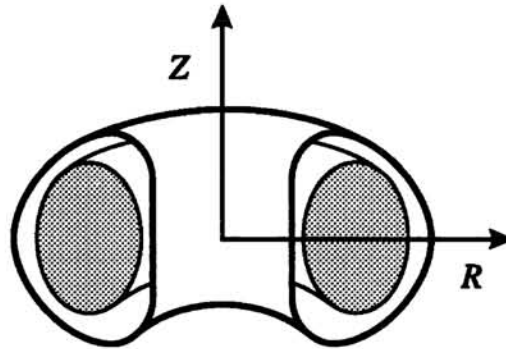

Figure 1: Schematic cross-section of a tokamak experiment showing the toroidal vacuum vessel (outer D-shaped curve) and plasma (shown shaded). Also shown are the radial ($R$) and vertical ($Z$) coordinates. To a good approximation, the tokamak can be regarded as axisymmetric about the $Z$-axis, and so the plasma boundary can be described by its cross-sectional shape at one particular toroidal location.

Unlike circular cross-section plasmas, highly non-circular shapes are more difficult to produce and to control accurately, since currents through several control coils must be adjusted simultaneously. Furthermore, during a typical plasma pulse, the shape must evolve, usually from some initial near-circular shape. Due to uncertainties in the current and pressure distributions within the plasma, the desired accuracy for plasma control can only be achieved by making real-time measurements of the position and shape of the boundary, and using error feedback to adjust the currents in the control coils.

The physics of the plasma equilibrium is determined by force balance between the

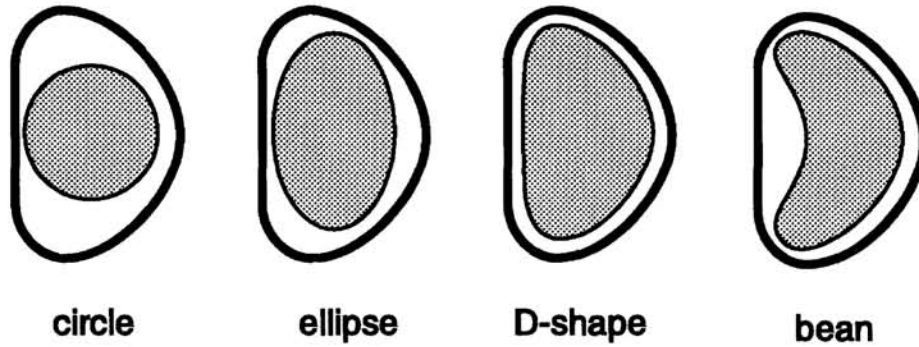

**circle**     **ellipse**     **D-shape**     **bean**

Figure 2: Cross-sections of the COMPASS vacuum vessel showing some examples of potential plasma shapes. The solid curve is the boundary of the vacuum vessel, and the plasma is shown by the shaded regions.

thermal pressure of the plasma and the pressure of the magnetic field, and is relatively well understood. Particular plasma configurations are described in terms of solutions of a non-linear partial differential equation called the Grad-Shafranov (GS) equation. Due to the non-linear nature of this equation, a general analytic solution is not possible. However, the GS equation can be solved by iterative numerical methods, with boundary conditions determined by currents flowing in the external control coils which surround the vacuum vessel. On the tokamak itself it is changes in these currents which are used to alter the position and cross-sectional shape of the plasma. Numerical solution of the GS equation represents the standard technique for post-shot analysis of the plasma, and is also the method used to generate the training dataset for the neural network, as described in the next section. However, this approach is computationally very intensive and is therefore unsuitable for feedback control purposes.

For real-time control it is necessary to have a fast (typically $\leq 50\mu sec.$) determination of the plasma boundary shape. This information can be extracted from a variety of diagnostic systems, the most important being local magnetic measurements taken at a number of points around the perimeter of the vacuum vessel. Most tokamaks have several tens or hundreds of small pick up coils located at carefully optimized points around the torus for this purpose. We shall represent these magnetic signals collectively as a vector $m$.

For a large class of equilibria, the plasma boundary can be reasonably well represented in terms of a simple parameterization, governed by an angle-like variable $\theta$, given by

$$
\begin{aligned}
R(\theta) &= R_0 + a\cos(\theta + \delta\sin\theta)\\
Z(\theta) &= Z_0 + a\kappa\sin\theta
\end{aligned}
\tag{1}
$$

where we have defined the following parameters

$R_0$    radial distance of the plasma center from the major axis of the torus,
$Z_0$    vertical distance of the plasma center from the torus midplane,
$a$      minor radius measured in the plane $Z = Z_0$,
$\kappa$ elongation,
$\delta$ triangularity.

We denote these parameters collectively by $y_k$. The basic problem which has to be addressed, therefore, is to find a representation for the (non-linear) mapping from the magnetic signals $m$ to the values of the geometrical parameters $y_k$, which can be implemented in suitable hardware for real-time control.

The conventional approach presently in use on many tokamaks involves approximating the mapping between the measured magnetic signals and the geometrical parameters by a single linear transformation. However, the intrinsic non-linearity of the mappings suggests that a representation in terms of feedforward neural networks should give significantly improved results (Lister and Schnurrenberger, 1991; Bishop *et al.*, 1992; Lagin *et al.*, 1993). Figure 3 shows a block diagram of the control loop for the neural network approach to tokamak equilibrium control.

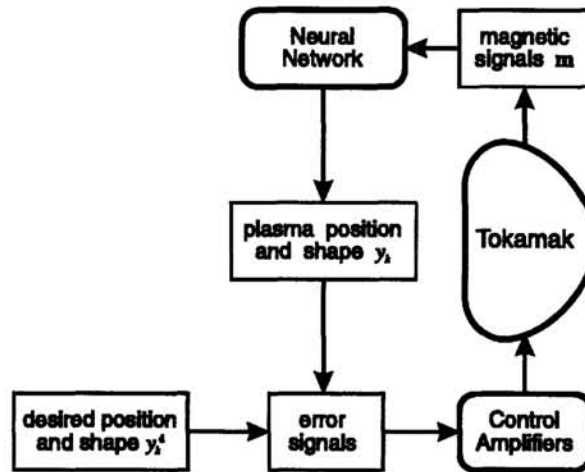

Figure 3: Block diagram of the control loop used for real-time feedback control of plasma position and shape.

## 2   SOFTWARE SIMULATION RESULTS

The dataset for training and testing the network was generated by numerical solution of the GS equation using a free-boundary equilibrium code. The data base currently consists of over 2,000 equilibria spanning the wide range of plasma positions and shapes available in COMPASS. Each equilibrium configuration takes several minutes to generate on a fast workstation. The boundary of each configuration is then fitted using the form in equation 1, so that the equilibria are labelled with the appropriate values of the shape parameters. Of the 120 magnetic signals available on COMPASS which could be used to provide inputs to the network, a

subset of 16 has been chosen using sequential forward selection based on a linear representation for the mapping (discussed below).

It is important to note that the transformation from magnetic signals to flux surface parameters involves an exact linear invariance. This follows from the fact that, if all of the currents are scaled by a constant factor, then the magnetic fields will be scaled by this factor, and the geometry of the plasma boundary will be unchanged. It is important to take advantage of this prior knowledge and to build it into the network structure, rather than force the network to learn it by example. We therefore normalize the vector $m$ of input signals to the network by dividing by a quantity proportional to the total plasma current. Note that this normalization has to be incorporated into the hardware implementation of the network, as will be discussed in Section 3.

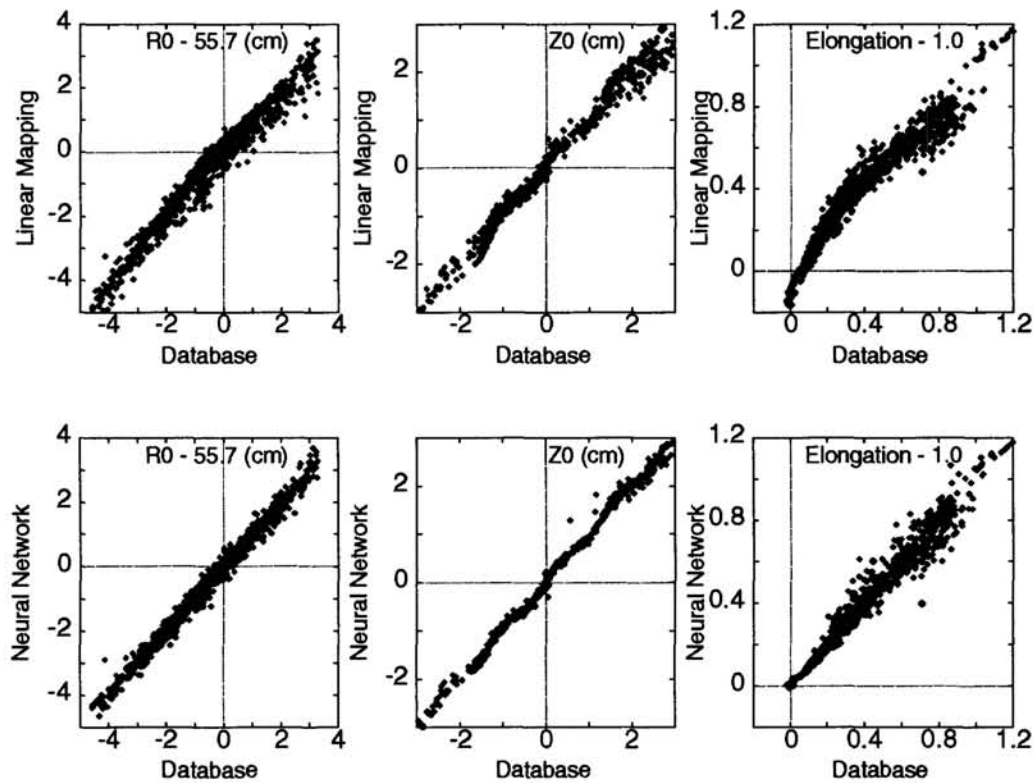

Figure 4: Plots of the values from the test set versus the values predicted by the linear mapping for the 3 equilibrium parameters, together with the corresponding plots for a neural network with 4 hidden units.

The results presented in this paper are based on a multilayer perceptron architecture having a single layer of hidden units with 'tanh' activation functions, and linear output units. Networks are trained by minimization of a sum-of-squares error using a standard conjugate gradients optimization algorithm, and the number of hidden

units is optimized by measuring performance with respect to an independent test set. Results from the neural network mapping are compared with those from the optimal linear mapping, that is the single linear transformation which minimizes the same sum-of-squares error as is used in the neural network training algorithm, as this represents the method currently used on a number of present day tokamaks.

Initial results were obtained on networks having 3 output units, corresponding to the values of vertical position $Z_0$, major radius $R_0$, and elongation $\kappa$; these being parameters which are of interest for real-time feedback control. The smallest normalized test set error of 11.7 is obtained from the network having 16 hidden units. By comparison, the optimal linear mapping gave a normalized test set error of 18.3. This represents a reduction in error of about 30% in going from the linear mapping to the neural network. Such an improvement, in the context of this application, is very significant.

For the experiments on real-time feedback control described in Section 4 the currently available hardware only permitted networks having 4 hidden units, and so we consider the results from this network in more detail. Figure 4 shows plots of the network predictions for various parameters versus the corresponding values from the test set portion of the database. Analogous plots for the optimal linear map predictions versus the database values are also shown. Comparison of the corresponding figures shows the improved predictive capability of the neural network, even for this sub-optimal network topology.

## 3  HARDWARE IMPLEMENTATION

The hardware implementation of the neural network must have a bandwidth of $\geq$ 20 kHz in order to cope with the fast timescales of the plasma evolution. It must also have an output precision of at least (the the analogue equivalent of) 8 bits in order to ensure that the final accuracy which is attainable will not be limited by the hardware system. We have chosen to develop a fully parallel custom implementation of the multilayer perceptron, based on analogue signal paths with digitally stored synaptic weights (Bishop *et al.*, 1993). A VME-based modular construction has been chosen as this allows flexibility in changing the network architecture, ease of loading network weights, and simplicity of data acquisition. Three separate types of card have been developed as follows:

- Combined 16-input buffer and signal normalizer.

  This provides an analogue hardware implementation of the input normalization described earlier.

- 16 × 4 matrix multiplier

  The synaptic weights are produced using 12 bit frequency-compensated multiplying DACs (digital to analogue converters) which can be configured to allow 4-quadrant multiplication of analogue signals by a digitally stored number.

- 4-channel sigmoid module

  There are many ways to produce a sigmoidal non-linearity, and we have opted for a solution using two transistors configured as a long-tailed-pair,

to generate a 'tanh' sigmoidal transfer characteristic. The principal draw-back of such an approach is the strong temperature sensitivity due to the appearance of temperature in the denominator of the exponential transistor transfer characteristic. An elegant solution to this problem has been found by exploiting a chip containing 5 transistors in close thermal contact. Two of the transistors form the long-tailed pair, one of the transistors is used as a heat source, and the remaining two transistors are used to measure temperature. External circuitry provides active thermal feedback control, and stability to changes in ambient temperature over the range 0°C to 50°C is found to be well within the acceptable range.

The complete network is constructed by mounting the appropriate combination of cards in a VME rack and configuring the network topology using front panel interconnections. The system includes extensive diagnostics, allowing voltages at all key points within the network to be monitored as a function of time via a series of multiplexed output channels.

# 4   RESULTS FROM REAL-TIME FEEDBACK CONTROL

Figure 5 shows the first results obtained from real-time control of the plasma in the COMPASS tokamak using neural networks. The evolution of the plasma elongation, under the control of the neural network, is plotted as a function of time during a plasma pulse. Here the desired elongation has been preprogrammed to follow a series of steps as a function of time. The remaining 2 network outputs (radial position $R_0$ and vertical position $Z_0$) were digitized for post-shot diagnosis, but were not used for real-time control. The solid curve shows the value of elongation given by the corresponding network output, and the dashed curve shows the post-shot reconstruction of the elongation obtained from a simple 'filament' code, which gives relatively rapid post-shot plasma shape reconstruction but with limited accuracy. The circles denote the elongation values given by the much more accurate reconstructions obtained from the full equilibrium code. The graph clearly shows the network generating the required elongation signal in close agreement with the reconstructed values. The typical residual error is of order 0.07 on elongation values up to around 1.5. Part of this error is attributable to residual offset in the integrators used to extract magnetic field information from the pick-up coils, and this is currently being corrected through modifications to the integrator design. An additional contribution to the error arises from the restricted number of hidden units available with the initial hardware configuration. While these results represent the first obtained using closed loop control, it is clear from earlier software modelling of larger network architectures (such as 32–16–4) that residual errors of order a few % should be attainable. The implementation of such larger networks is being persued, following the successes with the smaller system.

### Acknowledgements

We would like to thank Peter Cox, Jo Lister and Colin Roach for many useful discussions and technical contributions. This work was partially supported by the UK Department of Trade and Industry.

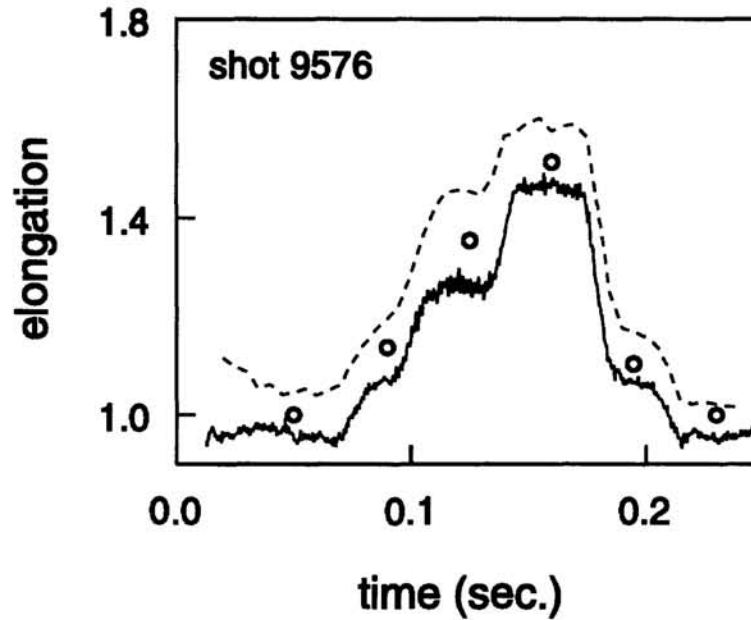

Figure 5: Plot of the plasma elongation $\kappa$ as a function of time during shot no. 9576 on the COMPASS tokamak, during which the elongation was being controlled in real-time by the neural network.

### References

Bishop C M, Cox P, Haynes P S, Roach C M, Smith M E U, Todd T N and Trotman D L, 1992. A neural network approach to tokamak equilibrium control. In *Neural Network Applications*, Ed. J G Taylor, Springer Verlag, 114–128.

Bishop C M, Haynes P S, Roach C M, Smith M E U, Todd T N, and Trotman D L. 1993. Hardware implementation of a neural network for plasma position control in COMPASS-D. In *Proceedings of the 17th. Symposium on Fusion Technology*, Rome, Italy. **2** 997–1001.

Lagin L, Bell R, Davis S, Eck T, Jardin S, Kessel C, Mcenerney J, Okabayashi M, Popyack J and Sauthoff N. 1993. Application of neural networks for real-time calculations of plasma equilibrium parameters for PBX-M, In *Proceedings of the 17th. Symposium on Fusion Technology*, Rome, Italy. **2** 1057–1061.

Lister J B and Schnurrenberger H. 1991. Fast non-linear extraction of plasma parameters using a neural network mapping. *Nuclear Fusion*. **31**, 1291–1300.
